# Fast Exact Inference with a Factored Model for Natural Language Parsing

**Dan Klein**
Department of Computer Science
Stanford University
Stanford, CA 94305-9040
klein@cs.stanford.edu

**Christopher D. Manning**
Department of Computer Science
Stanford University
Stanford, CA 94305-9040
manning@cs.stanford.edu

## Abstract

We present a novel generative model for natural language tree structures in which semantic (lexical dependency) and syntactic (PCFG) structures are scored with separate models. This factorization provides conceptual simplicity, straightforward opportunities for separately improving the component models, and a level of performance comparable to similar, non-factored models. Most importantly, unlike other modern parsing models, the factored model admits an extremely effective A* parsing algorithm, which enables efficient, exact inference.

## 1 Introduction

Syntactic structure has standardly been described in terms of categories (phrasal labels and word classes), with little mention of particular words. This is possible, since, with the exception of certain common function words, the acceptable syntactic configurations of a language are largely independent of the particular words that fill out a sentence. Conversely, for resolving the important attachment ambiguities of modifiers and arguments, lexical preferences are known to be very effective. Additionally, methods based only on key lexical dependencies have been shown to be very effective in choosing between valid syntactic forms [1]. Modern statistical parsers [2, 3] standardly use complex joint models of over both category labels and lexical items, where "everything is conditioned on everything" to the extent possible within the limits of data sparseness and finite computer memory. For example, the probability that a verb phrase will take a noun phrase object depends on the head word of the verb phrase. A VP headed by *acquired* will likely take an object, while a VP headed by *agreed* will likely not. There are certainly statistical interactions between syntactic and semantic structure, and, if deeper underlying variables of communication are not modeled, everything tends to be dependent on everything else in language [4]. However, the above considerations suggest that there might be considerable value in a factored model, which provides separate models of syntactic configurations and lexical dependencies, and then combines them to determine optimal parses. For example, under this view, we may know that *acquired* takes right dependents headed by nouns such as *company* or *division*, while *agreed* takes no noun-headed right dependents at all. If so, there is no need to explicitly model the phrasal selection on top of the lexical selection. Although we will show that such a model can indeed produce a high performance parser, we will focus particularly on how a factored model permits efficient, exact inference, rather than the approximate heuristic inference normally used in large statistical parsers.

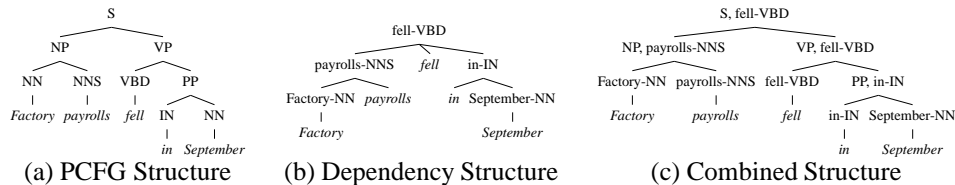

| (a) PCFG Structure | (b) Dependency Structure | (c) Combined Structure |

Figure 1: Three kinds of parse structures.

## 2 A Factored Model

Generative models for parsing typically model one of the kinds of structures shown in figure 1. Figure 1a is a plain phrase-structure tree $T$, which primarily models syntactic units, figure 1b is a dependency tree $D$, which primarily models word-to-word selectional affinities [5], and figure 1c is a lexicalized phrase-structure tree $L$, which carries both category and (part-of-speech tagged) head word information at each node.

A lexicalized tree can be viewed as the pair $L = (T, D)$ of a phrase structure tree $T$ and a dependency tree $D$. In this view, generative models over lexicalized trees, of the sort standard in lexicalized PCFG parsing [2, 3], can be regarded as assigning mass $P(T, D)$ to such pairs. To the extent that dependency and phrase structure need not be modeled jointly, we can factor our model as $P(T, D) = P(T)P(D)$: this approach is the basis of our proposed models, and its use is, to our knowledge, new. This factorization, of course, assigns mass to pairs which are incompatible, either because they do not generate the same terminal string or do not embody compatible bracketings. Therefore, the total mass assigned to valid structures will be less than one. We could imagine fixing this by renormalizing. For example, this situation fits into the product-of-experts framework [6], with one semantic expert and one syntactic expert that must agree on a single structure. However, since we are presently only interested in finding most-likely parses, no global renormalization constants need to be calculated.

Given the factorization $P(T, D) = P(T)P(D)$, rather than engineering a single complex combined model, we can instead build two simpler sub-models. We show that the combination of even quite simple "off the shelf" implementations of the two sub-models can provide decent parsing performance. Further, the modularity afforded by the factorization makes it much easier to extend and optimize the individual components. We illustrate this by building improved versions of both sub-models, but we believe that there is room for further optimization.

Concretely, we used the following sub-models. For $P(T)$, we used successively more accurate PCFGs. The simplest, PCFG-BASIC, used the raw treebank grammar, with nonterminals and rewrites taken directly from the training trees [7]. In this model, nodes rewrite atomically, in a top-down manner, in only the ways observed in the training data. For improved models of $P(T)$, tree nodes' labels were annotated with various contextual markers. In PCFG-PA, each node was marked with its parent's label as in [8]. It is now well known that such annotation improves the accuracy of PCFG parsing by weakening the PCFG independence assumptions. For example, the NP in figure 1a would actually have been labeled NP^S. Since the counts were not fragmented by head word or head tag, we were able to directly use the MLE parameters, without smoothing.[1] The best PCFG model, PCFG-LING, involved selective parent splitting, order-2 rule markovization (similar to [2, 3]), and linguistically-derived feature splits.[2]

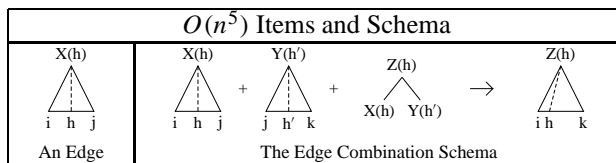

Figure 2: Edges and the edge combination schema for an $O(n^5)$ lexicalized tabular parser.

Models of $P(D)$ were lexical dependency models, which deal with tagged words: pairs $\langle w, t \rangle$. First the head $\langle w_h, t_h \rangle$ of a constituent is generated, then successive right dependents $\langle w_d, t_d \rangle$ until a STOP token $\diamond$ is generated, then successive left dependents until $\diamond$ is generated again. For example, in figure 1, first we choose *fell*-VBD as the head of the sentence. Then, we generate *in*-IN to the right, which then generates *September*-NN to the right, which generates $\diamond$ on both sides. We then return to *in*-IN, generate $\diamond$ to the right, and so on.

The dependency models required smoothing, as the word-word dependency data is very sparse. In our basic model, DEP-BASIC, we generate a dependent conditioned on the head and direction, using a mixture of two generation paths: a head can select a specific argument word, or a head can select only an argument tag. For head selection of words, there is a prior distribution over dependents taken by the head's tag, for example, left dependents taken by past tense verbs: $P(w_d, t_d | t_h, dir) = \text{count}(w_d, t_d, t_h, dir)/\text{count}(t_h, dir)$. Observations of bilexical pairs are taken against this prior, with some prior strength $\kappa$:

$$P(w_d, t_d | w_h, t_h, dir) = \frac{\text{count}(w_d, t_d, w_h, t_h, dir) + \kappa P(w_d, t_d | t_h, dir)}{\text{count}(w_h, t_h, dir) + \kappa}$$

This model can capture bilexical selection, such as the affinity between *payrolls* and *fell*. Alternately, the dependent can have only its tag selected, and then the word is generated independently: $P(w_d, t_d | w_h, t_h, dir) = P(w_d | t_d)P(t_d | w_h, t_h, dir)$. The estimates for $P(t_d | w_h, t_h, dir)$ are similar to the above. These two mixture components are then linearly interpolated, giving just two prior strengths and a mixing weight to be estimated on held-out data.

In the enhanced dependency model, DEP-VAL, we condition not only on direction, but also on distance and valence. The decision of whether to generate $\diamond$ is conditioned on one of five values of distance between the head and the generation point: zero, one, 2–5, 6–10, and 11+. If we decide to generate a non-$\diamond$ dependent, the actual choice of dependent is sensitive only to whether the distance is zero or not. That is, we model only zero/non-zero valence. Note that this is (intentionally) very similar to the generative model of [2] in broad structure, but substantially less complex.

At this point, one might wonder what has been gained. By factoring the semantic and syntactic models, we have certainly simplified both (and fragmented the data less), but there are always simpler models, and researchers have adopted complex ones because of their parsing accuracy. In the remainder of the paper, we demonstrate the three primary benefits of our model: a fast, exact parsing algorithm; parsing accuracy comparable to non-factored models; and useful modularity which permits easy extensibility.

---

several subtypes, conjunctions were split into contrastive and other occurrences, and the word *not* was given a unique tag. In all models, unknown words were modeled using only the MLE of $P(\text{tag}|\text{unknown})$ with ML estimates for the reserved mass per tag. Selective splitting was done using an information-gain like criterion.

# 3 An A* Parser

In this section, we outline an efficient algorithm for finding the Viterbi, or most probable, parse for a given terminal sequence in our factored lexicalized model. The naive approach to lexicalized PCFG parsing is to act as if the lexicalized PCFG is simply a large nonlexical PCFG, with many more symbols than its nonlexicalized PCFG backbone. For example, while the original PCFG might have a symbol NP, the lexicalized one has a symbol NP-$x$ for every possible head $x$ in the vocabulary. Further, rules like S → NP VP become a family of rules S-$x$ → NP-$y$ VP-$x$.[3] Within a dynamic program, the core parse item in this case is the *edge*, shown in figure 2, which is specified by its start, end, root symbol, and head position.[4] Adjacent edges combine to form larger edges, as in the top of figure 2. There are $O(n^3)$ edges, and two edges are potentially compatible whenever the left one ends where the right one starts. Therefore, there are $O(n^5)$ such combinations to check, giving an $O(n^5)$ dynamic program.[5]

The core of our parsing algorithm is a tabular agenda-based parser, using the $O(n^5)$ schema above. The novelty is in the choice of agenda priority, where we exploit the rapid parsing algorithms available for the sub-models to speed up the otherwise impractical combined parse. Our choice of priority also guarantees optimality, in the sense that when the goal edge is removed, its most probable parse is known exactly. Other lexicalized parsers accelerate parsing in ways that destroy this optimality guarantee. The top-level procedure is given in figure 3. First, we parse exhaustively with the two sub-models, not to find complete parses, but to find best *outside scores* for each edge $e$. An outside score is the score of the best parse structure which starts at the goal and includes $e$, the words before it, and the words after it, as depicted in figure 3. Outside scores are a Viterbi analog of the standard outside probabilities given by the inside-outside algorithm [11]. For the syntactic model, $P(T)$, well-known cubic PCFG parsing algorithms are easily adapted to find outside scores. For the semantic model, $P(D)$, there are several presentations of cubic dependency parsing algorithms, including [9] and [12]. These can also be adapted to produce outside scores in cubic time, though since their basic data structures are not edges, there is some subtlety. For space reasons, we omit the details of these phases.

An agenda-based parser tracks all edges that have been constructed at a given time. When an edge is first constructed, it is put on an agenda, which is a priority queue indexed by some score for that node. The agenda is a holding area for edges which have been built in at least one way, but which have not yet been used in the construction of other edges. The core cycle of the parser is to remove the highest-priority edge from the agenda, and act on it according to the edge combination schema, combining it with any previously removed, compatible edges. This much is common to many parsers; agenda-based parsers primarily differ in their choice of edge priority. If the best known inside score for an edge is used as a priority, then the parser will be optimal. In particular, when the goal edge is removed, its score will correspond the most likely parse. The proof is a generalization of the proof of Dijkstra's algorithm (uniform-cost search), and is omitted for space reasons

1. Extract the PCFG sub-model and set up the PCFG parser.
2. Use the PCFG parser to find outside scores $\alpha_{PCFG}(e)$ for each edge.
3. Extract the dependency sub-model and set up the dependency parser.
4. Use the dependency parser to find outside scores $\alpha_{DEP}(e)$ for each edge.
5. Combine PCFG and dependency sub-models into the lexicalized model.
6. Form the combined outside estimate $a(e) = \alpha_{PCFG}(e) + \alpha_{DEP}(e)$
7. Use the lexicalized A* parser, with $a(e)$ as an A* estimate of $\alpha(e)$

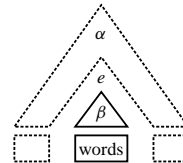

Figure 3: The top-level algorithm and an illustration of inside and outside scores.

| PCFG Model | Precision | Recall | $F_1$ | Exact Match |
|---|---|---|---|---|
| PCFG-BASIC | 75.3 | 70.2 | 72.7 | 11.0 |
| PCFG-PA | 78.4 | 76.9 | 77.7 | 18.5 |
| PCFG-LING | 83.7 | 82.1 | 82.9 | 25.7 |

(a) The PCFG Model

| Dependency Model | Dependency Acc |
|---|---|
| DEP-BASIC | 76.3 |
| DEP-VAL | 85.0 |

(b) The Dependency Model

Figure 4: Performance of the sub-models alone.

(but given in [13]). However, removing edges by inside score is not practical (see section 4 for an empirical demonstration), because all small edges end up having better scores than any large edges. Luckily, the optimality of the algorithm remains if, rather than removing items from the agenda by their best inside scores, we add to those scores any optimistic (admissible) estimate of the cost to complete a parse using that item. The proof of this is a generalization of the proof of the optimality of A* search.

To our knowledge, no way of generating effective, admissible A* estimates for lexicalized parsing has previously been proposed.[6] However, because of the factored structure of our model, we can use the results of the sub-models' parses to give us quite sharp A* estimates. Say we want to know the outside score of an edge $e$. That score will be the score $\alpha(T_e, D_e)$ (a logprobability) of a certain structure $(T_e, D_e)$ outside of $e$, where $T_e$ and $D_e$ are a compatible pair. From the initial phases, we know the exact scores of the overall best $T'_e$ and the best $D'_e$ which can occur outside of $e$, though of course it may well be that $T'_e$ and $D'_e$ are not compatible. However, $\alpha_{PCFG}(T_e) \leq \alpha_{PCFG}(T'_e)$ and $\alpha_{DEP}(D_e) \leq \alpha_{DEP}(D'_e)$, and so $\alpha(T_e, D_e) = \alpha_{PCFG}(T_e) + \alpha_{DEP}(D_e) \leq \alpha_{PCFG}(T'_e) + \alpha_{DEP}(D'_e)$. Therefore, we can use the sum of the sub-models' outside scores, $a(e) = \alpha_{PCFG}(T'_e) + \alpha_{DEP}(D'_e)$, as an upper bound on the outside score for the combined model. Since it is reasonable to assume that the two models will be broadly compatible and will generally prefer similar structures, this should create a sharp A* estimate, and greatly reduce the work needed to find the goal parse. We give empirical evidence of this in section 4.

## 4 Empirical Performance

In this section, we demonstrate that (i) the factored model's parsing performance is comparable to non-factored models which use similar features, (ii) there is an advantage to exact inference, and (iii) the A* savings are substantial. First, we give parsing figures on the standard Penn treebank parsing task. We trained the two sub-models, separately, on sections 02–21 of the WSJ section of the treebank. The numbers reported here are the result of then testing on section 23 (length $\leq 40$). The treebank only supplies node labels (like NP) and

| PCFG Model | Dependency Model | Precision | Recall | $F_1$ | Exact Match | Dependency Acc |
|---|---|---|---|---|---|---|
| PCFG-BASIC | DEP-BASIC | 80.1 | 78.2 | 79.1 | 16.7 | 87.2 |
| PCFG-BASIC | DEP-VAL | 82.5 | 81.5 | 82.0 | 17.7 | 89.2 |
| PCFG-PA | DEP-BASIC | 82.1 | 82.2 | 82.1 | 23.7 | 88.0 |
| PCFG-PA | DEP-VAL | 84.0 | 85.0 | 84.5 | 24.8 | 89.7 |
| PCFG-LING | DEP-BASIC | 85.4 | 84.8 | 85.1 | 30.4 | 90.3 |
| PCFG-LING | DEP-VAL | 86.6 | 86.8 | 86.7 | 32.1 | 91.0 |

| PCFG Model | Dependency Model | Thresholded? | $F_1$ | Exact Match | Dependency Acc |
|---|---|---|---|---|---|
| PCFG-LING | DEP-VAL | No | 86.7 | 32.1 | 91.0 |
| PCFG-LING | DEP-VAL | Yes | 86.5 | 31.9 | 90.8 |

Figure 5: The combined model, with various sub-models, and with/without thresholding.

does not contain head information. Heads were calculated for each node according to the deterministic rules given in [2]. These rules are broadly correct, but not perfect.

We effectively have three parsers: the PCFG (sub-)parser, which produces nonlexical phrase structures like figure 1a, the dependency (sub-)parser, which produces dependency structures like figure 1b, and the combination parser, which produces lexicalized phrase structures like figure 1c. The outputs of the combination parser can also be projected down to either nonlexical phrase structures or dependency structures. We score the output of our parsers in two ways. First, the phrase structure of the PCFG and combination parsers can be compared to the treebank parses. The parsing measures standardly used for this task are labeled precision and recall.[7] We also report $F_1$, the harmonic mean of these two quantities. Second, for the dependency and combination parsers, we can score the dependency structures. A dependency structure $D$ is viewed as a set of head-dependent pairs $\langle h, d \rangle$, with an extra dependency $\langle root, x \rangle$ where $root$ is a special symbol and $x$ is the head of the sentence. Although the dependency model generates part-of-speech tags as well, these are ignored for dependency accuracy. Punctuation is not scored. Since all dependency structures over $n$ non-punctuation terminals contain $n$ dependencies ($n - 1$ plus the root dependency), we report only accuracy, which is identical to both precision and recall. It should be stressed that the "correct" dependency structures, though generally correct, are generated from the PCFG structures by linguistically motivated, but automatic and only heuristic rules.

Figure 4 shows the relevant scores for the various PCFG and dependency parsers alone.[8] The valence model increases the dependency model's accuracy from 76.3% to 85.0%, and each successive enhancement improves the $F_1$ of the PCFG models, from 72.7% to 77.7% to 82.9%. The combination parser's performance is given in figure 5. As each individual model is improved, the combination $F_1$ is also improved, from 79.1% with the pair of basic models to 86.7% with the pair of top models. The dependency accuracy also goes up: from 87.2% to 91.0%. Note, however, that even the pair of basic models has a combined dependency accuracy higher than the enhanced dependency model alone, and the top three have combined $F_1$ better than the best PCFG model alone. For the top pair, figure 6c illustrates the relative $F_1$ of the combination parser to the PCFG component alone, showing the unsurprising trend that the addition of the dependency model helps more for longer sentences, which, on average, contain more attachment ambiguity. The top $F_1$ of 86.7% is greater than that of the lexicalized parsers presented in [15, 16], but less than that of the newer, more complex, parsers presented in [3, 2], which reach as high as 90.1% $F_1$.

[7]A tree $T$ is viewed as a set of constituents $c(T)$. Constituents in the correct and the proposed tree must have the same start, end, and label to be considered identical. For this measure, the lexical heads of nodes are irrelevant. The actual measures used are detailed in [15], and involve minor normalizations like the removal of punctuation in the comparison.

[8]The dependency model is sensitive to any preterminal annotation (tag splitting) done by the PCFG model. The actual value of DEP-VAL shown corresponds to PCFG-LING.

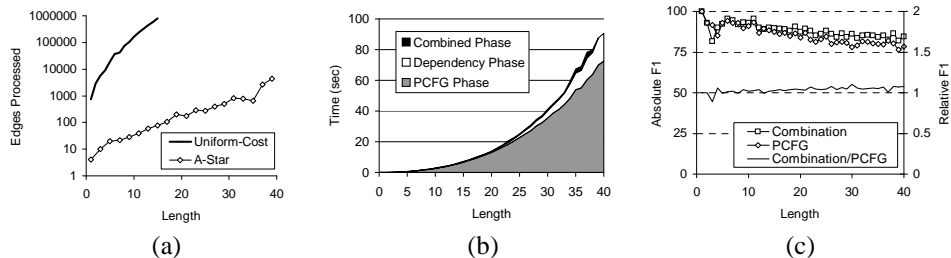

Figure 6: (a) A* effectiveness measured by edges expanded, (b) time spent on each phase, and (c) relative $F_1$, all shown as sentence length increases.

However, it is worth pointing out that these higher-accuracy parsers incorporate many finely wrought enhancements which could presumably be extracted and applied to benefit our individual models.[9]

The primary goal of this paper is not to present a maximally tuned parser, but to demonstrate a method for fast, exact inference usable in parsing. Given the impracticality of exact inference for standard parsers, a common strategy is to take a PCFG backbone, extract a set of top parses, either the top $k$ or all parses within a score threshold of the top parse, and rerank them [3, 17]. This pruning is done for efficiency; the question is whether it is hurting accuracy. That is, would exact inference be preferable? Figure 5 shows the result of parsing with our combined model, using the best model pair, but with the A* estimates altered to block parses whose PCFG projection had a score further than a threshold $\delta = 2$ in log-probability from the best PCFG-only parse. Both bracket $F_1$ and exact-match rate are lower for the thresholded parses, which we take as an argument for exact inference.[10]

We conclude with data on the effectiveness of the A* method. Figure 6a shows the average number of edges extracted from the agenda as sentence length increases. Numbers both with and without using the A* estimate are shown. Clearly, the uniform-cost version of the parser is dramatically less efficient; by sentence length 15 it extracts over 800K edges, while even at length 40 the A* heuristics are so effective that only around 2K edges are extracted. At length 10, the average number is less than 80, and the fraction of edges not suppressed is better than 1/10K (and improves as sentence length increases). To explain this effectiveness, we suggest that the combined parsing phase is really only figuring out how to reconcile the two models' preferences.[11] The A* estimates were so effective that even with our object-heavy Java implementation of the combined parser, total parse time was dominated by the initial, array-based PCFG phase (see figure 6b).[12]

# 5 Conclusion

The framework of factored models over lexicalized trees has several advantages. It is conceptually simple, and modularizes the model design and estimation problems. The concrete model presented performs comparably to other, more complex, non-exact models proposed, and can be easily extended in the ways that other parser models have been. Most importantly, it admits a novel A* parsing approach which allows fast, exact inference of the most probable parse.

**Acknowledgements.** We would like to thank Lillian Lee, Fernando Pereira, and Joshua Goodman for advice and discussion about this work. This paper is based on work supported by the National Science Foundation (NSF) under Grant No. IIS-0085896, by the Advanced Research and Development Activity (ARDA)'s Advanced Question Answering for Intelligence (AQUAINT) Program, by an NSF Graduate Fellowship to the first author, and by an IBM Faculty Partnership Award to the second author.

## Footnotes

[1]This is not to say that smoothing would not improve performance, but to underscore how the factored model encounters less sparsity problems than a joint model.

[2]Infinitive VPs, possessive NPs, and gapped Ss were marked, the preposition tag was split into

[3]The score of such a rule in the factored model would be the PCFG score for S → NP VP, combined with the score for $x$ taking $y$ as a dependent and the left and right STOP scores for $y$.

[4]The head position variable often, as in our case, also specifies the head's tag.

[5]Eisner and Satta [9] propose a clever $O(n^4)$ modification which separates this process into two steps by introducing an intermediate object. However, even the $O(n^4)$ formulation is impractical for exhaustive parsing with broad-coverage, lexicalized treebank grammars. There are several reasons for this: the constant factor due to the grammar is huge (these grammars often contain tens of thousands of rules once binarized), and larger sentences are more likely to contain structures which unlock increasingly large regions of the grammar ([10] describes how this can cause the sentence length to leak into terms which are analyzed as constant, leading to empirical growth far faster than the predicted bounds). We did implement a version of this parser using the $O(n^4)$ formulation of [9], but, because of the effectiveness of the A* estimate, it was only marginally faster; see section 4.

[6]The basic idea of changing edge priorities to more effectively guide parser work is standardly used, and other authors have made very effective use of inadmissible estimates. [2] uses extensive probabilistic pruning – this amounts to giving pruned edges infinitely low priority. Absolute pruning can, and does, prevent the most likely parse from being returned at all. [14] removes edges in order of estimates of their correctness. This, too, may result in the first parse found not being the most likely parse, but it has another more subtle drawback: if we hold back an edge $e$ for too long, we may use $e$ to build another edge $f$ in a new, better way. If $f$ has already been used to construct larger edges, we must then propagate its new score upwards (which can trigger still further propagation).

[9]For example, the dependency distance function of [2] registers punctuation and verb counts, and both smooth the PCFG production probabilities, which could improve the PCFG grammar.

[10]While pruning typically buys speed at the expense of some accuracy (see also, e.g., [2]), pruning can also sometimes improve $F_1$: Charniak et al. [14] find that pruning based on estimates for $P(e|s)$ raises accuracy slightly, for a non-lexicalized PCFG. As they note, their pruning metric seems to mimic Goodman's maximum-constituents parsing [18], which maximizes the expected number of correct nodes rather than the likelihood of the entire parse. In any case, we see it as valuable to have an exact parser with which these types of questions can be investigated at all for lexicalized parsing.

[11]Note that the uniform-cost parser does enough work to exploit the shared structure of the dynamic program, and therefore edge counts appear to grow polynomially. However, the A* parser does so little work that there is minimal structure-sharing. Its edge counts therefore appear to grow *exponentially* over these sentence lengths, just like a non-dynamic-programming parser's would. With much longer sentences, or a less efficient estimate, the polynomial behavior would reappear.

[12]The average time to parse a sentence with the best model on a 750MHz Pentium III with 2GB RAM was: for 20 words, PCFG 13 sec, dependencies 0.6 sec, combination 0.3 sec; 40 words, PCFG 72 sec, dependencies 18 sec, combination 1.6 sec.

# References

[1] D. Hindle and M. Rooth. Structural ambiguity and lexical relations. *Computational Linguistics*, 19(1):103–120, 1993.

[2] M. Collins. *Head-Driven Statistical Models for Natural Language Parsing*. PhD thesis, University of Pennsylvania, 1999.

[3] E. Charniak. A maximum-entropy-inspired parser. *NAACL 1*, pp. 132–139, 2000.

[4] R. Bod. What is the minimal set of fragments that achieves maximal parse accuracy? *ACL 39*, pp. 66–73, 2001.

[5] I. A. Mel'čuk. *Dependency Syntax: theory and practice*. State University of New York Press, Albany, NY, 1988.

[6] G. E. Hinton. Training products of experts by minimizing contrastive divergence. Technical Report GCNU TR 2000-004, GCNU, University College London, 2000.

[7] E. Charniak. Tree-bank grammars. *Proceedings of the Thirteenth National Conference on Artificial Intelligence (AAAI '96)*, pp. 1031–1036, 1996.

[8] M. Johnson. PCFG models of linguistic tree representations. *Computational Linguistics*, 24(4):613–632, 1998.

[9] J. Eisner and G. Satta. Efficient parsing for bilexical context-free grammars and head-automaton grammars. *ACL 37*, pp. 457–464, 1999.

[10] D. Klein and C. D. Manning. Parsing with treebank grammars: Empirical bounds, theoretical models, and the structure of the Penn treebank. *ACL 39/EACL 10*, pp. 330–337, 2001.

[11] J. K. Baker. Trainable grammars for speech recognition. D. H. Klatt and J. J. Wolf, editors, *Speech Communication Papers for the 97th Meeting of the Acoustical Society of America*, pp. 547–550, 1979.

[12] J. Lafferty, D. Sleator, and D. Temperley. Grammatical trigrams: A probabilistic model of link grammar. *Proc. AAAI Fall Symposium on Probabilistic Approaches to Natural Language*, 1992.

[13] D. Klein and C. D. Manning. Parsing and hypergraphs. *Proceedings of the 7th International Workshop on Parsing Technologies (IWPT-2001)*, 2001.

[14] E. Charniak, S. Goldwater, and M. Johnson. Edge-based best-first chart parsing. *Proceedings of the Sixth Workshop on Very Large Corpora*, pp. 127–133, 1998.

[15] D. M. Magerman. Statistical decision-tree models for parsing. *ACL 33*, pp. 276–283, 1995.

[16] M. J. Collins. A new statistical parser based on bigram lexical dependencies. *ACL 34*, pp. 184–191, 1996.

[17] M. Collins. Discriminative reranking for natural language parsing. *ICML 17*, pp. 175–182, 2000.

[18] J. Goodman. Parsing algorithms and metrics. *ACL 34*, pp. 177–183, 1996.
